# Who's Doing What: Joint Modeling of Names and Verbs for Simultaneous Face and Pose Annotation

**Luo Jie**
Idiap and EPF Lausanne
jluo@idiap.ch

**Barbara Caputo**
Idiap Research Institute
bcaputo@idiap.ch

**Vittorio Ferrari**
ETH Zurich
ferrari@vision.ee.ethz.ch

## Abstract

Given a corpus of news items consisting of images accompanied by text captions, we want to find out "who's doing what", i.e. associate names and action verbs in the captions to the face and body pose of the persons in the images. We present a joint model for simultaneously solving the image-caption correspondences and learning visual appearance models for the face and pose classes occurring in the corpus. These models can then be used to recognize people and actions in novel images without captions. We demonstrate experimentally that our joint 'face and pose' model solves the correspondence problem better than earlier models covering only the face, and that it can perform recognition of new uncaptioned images.

## 1 Introduction

A huge amount of images with accompanying text captions are available on the Internet. Websites selling various items such as houses and clothing provide photographs of their products along with concise descriptions. Online newspapers [1] have pictures illustrating events and comment them in the caption. These news websites are very popular because people are interested in other people, especially if they are famous (figure 1). Exploiting the associations between images and text hidden in this wealth of data can lead to a virtually infinite source of annotations from which to learn visual models without explicit manual intervention.

The learned models could then be used in a variety of Computer Vision applications, including face recognition, image search engines, and to annotate new images for which no caption is available. Moreover, recovering image-text associations is useful for auto-annotating a closed corpus of data, e.g. for users of news website to see "who's in the picture" [6], or to search for images where a certain person does a certain thing.

Previous works on news items has focused on associating names in the captions to faces in the images [5, 6, 16, 21]. This is difficult due to the *correspondence ambiguity* problem: multiple persons appear in the image and the caption. Moreover, persons in the image are not always mentioned in the caption, and not all names in the caption appear in the image. The techniques tackle the correspondence problem by exploiting the fact that different images show different combinations of persons. As a result, these methods work well for frequently occurring persons (typical for famous people) appearing in dataset with thousands of news items.

In this paper we propose to go beyond the above works, by modeling both *names* and *action verbs* jointly. These correspond to *faces* and *body poses* in the images (figure 3). The connections between the subject (name) and verb in a caption can be found by well established language analysis techniques [1, 8]. Essentially, by considering the subject-verb language construct, we generalize the "who's in the picture" line of works to "who's doing what". We present a new generative model where the observed variables are names and verbs in the caption as well as detected persons in the image. The image-caption correspondences are carried by latent variables, while the visual appearance of face and pose classes corresponding to different names and verbs are model parameters. During learning, we simultaneously solve for the correspondence and learn the appearance models.

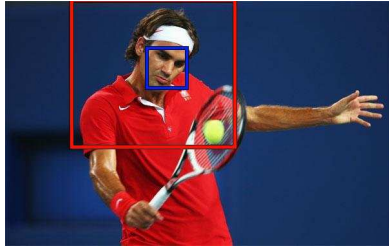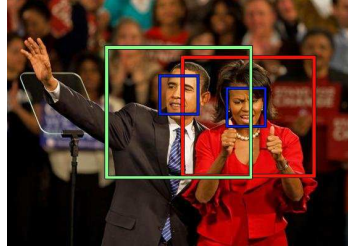

*(a) Four sets ...* **Roger Federer** *prepares to* **hit a backhand** *in a quarter-final match with* **Andy Roddick** *at the US Open.*

*(b) US Democratic presidential candidate Senator* **Barack Obama** **waves** *to supporters together with his wife* **Michelle Obama** **standing** *beside him at his North Carolina and Indiana primary election night rally in Raleigh.*

Figure 1: Examples of image-caption pairs in our dataset. The face and upper body of the persons in the image are marked by bounding-boxes. We stress a caption might contain names and/or verbs not visible in the image, and vice versa.

In our joint model, the correspondence ambiguity is reduced because the face and pose information *help each other*. For example, in figure 1b, knowing what 'waves' means would reveal who of the two imaged persons is Obama. The other way around, knowing who is Obama would deliver a visual example for the 'waving' pose.

We show experimentally that (i) our joint 'face and pose' model solves the correspondence problem better than simpler models covering either face or pose alone; (ii) the learned model can be used to effectively annotate new images with or without captions; (iii) our model with face alone performs better than the existing face-only methods based on Gaussian mixture appearance models.

**Related works.** This paper is most closely related to works on associating names and faces, which we discussed above. There exist also works on associating nouns to image regions [2, 3, 10], starting from images annotated with a list of nouns indicating the objects it contains (typical datasets contain natural scenes and objects such as 'water' and 'tiger'). A recent work in this line is that of Gupta and Davis [17], who model prepositions in addition to nouns (e.g. 'bear in water', 'car on street'). To the best of our knowledge, ours is the first work on jointly modeling names and verbs.

## 2 Generative model for faces and body poses

The news item corpus used to train our face and pose model consists of still images of person(s) performing some action(s). Each image is annotated with a caption describing "who's doing what" in the image (figure 1). Some names from the caption might not appear in the image, and vice-versa some imaged persons might not be mentioned in the caption. The basic units in our model are persons in the image, consisting of their face and upper body. Our system automatically detects them by bounding-boxes in the image using a face detector [23] and an upper body detector [14]. In the rest of the paper, we say "person" to indicate a detected face and the upper body associated with it (including false positive detections). A face and an upper-body are considered to belong to the same person if the face lies near the center of the upper body bounding-box. For each person, we obtain a pose estimate using [11] (figure 3(right)). In addition to these image features, we use a language parser [1, 8] to extract a set of name-verb pairs from each caption. Our goals are to: (i) associate the persons in the images to the name-verb pairs in the captions, and (ii) learn visual appearance models corresponding to names and verbs. These can then be used for recognition on new images with or without caption. Learning in our model can be seen as a constrained clustering problem [4, 24, 25].

### 2.1 Generative model

We start by describing how our generative model explains the image-caption data (figure 2). The notation is summarized in Table I. Suppose we have a collection of documents $\boldsymbol{D} = \{D^1, \ldots, D^M\}$ with each document $D^i$ consisting of an image $I^i$ and its caption $C^i$. These captions implicitly provide the labels of the person(s)' name(s) and pose(s) in the corresponding images. For each caption $C^i$, we consider only the name-verb pairs $\boldsymbol{n}^i$ returned by a language parser [1, 8] and ignore other words. We make the same assumptions as for the name-face problem [5, 6, 16, 21] that the labels can only come from the name-verb pairs in the captions or *null* (for persons not mentioned in the caption). Based on this, we generate the set of all possible assignments $\boldsymbol{A}^i$ from the $\boldsymbol{n}^i$ in

| | |
|---|---|
| $M$: Number of documents in $\boldsymbol{D}$ (image-caption pairs) | $\boldsymbol{D} = \{D^i\}_{i=1}^{i=M} = \{I^i, C^i\}_{i=1}^{i=M}$ |
| $P^i$: Number of detected persons in image $I^i$ | $I^{i,p}$: $p$th person in image $I^i$ |
| $W^i$: Number of name-verb pairs in caption $C^i$ | $I^{i,p} = (I_{\text{face}}^{i,p}, I_{\text{pose}}^{i,p})$ |
| $\boldsymbol{Y}$: Latent variables encoding the true assignments | |
| $Y^i$: $Y^i = (y^{i,1}, \dots, y^{i,P^i})$, $y^{i,p}$ is the assignment of the $p$th person in $i$th image | |
| $\boldsymbol{A}^i$: Set of possible assignments for document $i$ | $\boldsymbol{A}^i = \{a_1^i, \dots, a_{L^i}^i\}$ |
| $L^i$: Number of possible assignments for document $D^i$ | |
| $a_l^i$: $l$th assignment $a_l^i = \{a_l^{i,1}, \dots, a_l^{i,P^i}\}$, where $a_l^{i,p}$ is the label for the $p$th person | |
| $\boldsymbol{\Theta}$: Appearance models for face and pose classes | $\boldsymbol{\Theta} = (\theta_{\text{name}}, \theta_{\text{verb}})$ |
| $V$: Number of different verbs | $\theta_{\text{verb}} = (\theta_{\text{verb}}^1, \dots, \theta_{\text{verb}}^V, \beta_{\text{verb}})$ |
| $U$: Number of different names | $\theta_{\text{name}} = (\theta_{\text{name}}^1, \dots, \theta_{\text{name}}^U, \beta_{\text{name}})$ |
| $\theta^k$: Sets of class representative vectors for class $k$ | $\mu_r^k$: a representative vector for class $k$ |
| $\theta_{\text{verb}}^v = \{\mu_{\text{pose}}^{v,1}, \dots, \mu_{\text{pose}}^{v,\text{R}^v}\}$ | $\theta_{\text{name}}^u = \{\mu_{\text{face}}^{u,1}, \dots, \mu_{\text{face}}^{u,\text{R}^u}\}$ |

Table I: The mathematical notation used in the paper

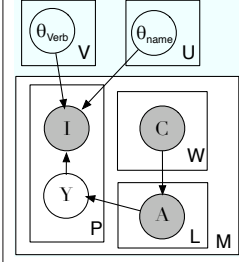

Figure 2: Graphical plate representation of the generative model.

$C^i$ (see section 2.4 for details). Hence, we replace the captions by the sets of possible assignments $\mathcal{A} = \{\boldsymbol{A}^1, \dots, \boldsymbol{A}^M\}$. Let $\boldsymbol{Y} = \{Y^1, \dots, Y^M\}$ be latent variables encoding the true assignments (i.e. name/verb labels for the faces/poses), and $Y^i = (y^{i,1}, \dots, y^{i,P^i})$ be the assignment for the $P^i$ persons in the $i$th image. Each $y^{i,p} = (y_{\text{face}}^{i,p}, y_{\text{pose}}^{i,p})$ is a pair of indices defining the assignment of a person's face to a name and pose to a verb. These take on values from the set of name indices $\{1, \dots, \text{U}, \textit{null}\}$, and verb indices $\{1, \dots, \text{V}, \textit{null}\}$. $N/V$ is the number of different names/verbs over all the captions and *null* represents unknown names/verbs and false positive person detections.

**Document collection likelihood.** Assuming independence between documents, the likelihood of the whole document collection is

$$P(\boldsymbol{I}, \boldsymbol{Y}, \mathcal{A}|\boldsymbol{\Theta}) = \prod_{i=1}^{M} P(I^i, Y^i, \boldsymbol{A}^i|\boldsymbol{\Theta}) = \prod_{i=1}^{M} P(I^i|Y^i, \boldsymbol{A}^i, \boldsymbol{\Theta}) P(Y^i|\boldsymbol{A}^i, \boldsymbol{\Theta}) P(\boldsymbol{A}^i|\boldsymbol{\Theta}) \quad (1)$$

where $\boldsymbol{\Theta}$ are the model parameters explaining the visual appearance of the persons' faces and poses in the images. Therefore, equation (1) can be written as $\prod P(I^i|Y^i, \boldsymbol{\Theta}) P(Y^i|\boldsymbol{A}^i) P(\boldsymbol{A}^i)$. The goal of learning is to find the parameters $\boldsymbol{\Theta}$ and the labels $\boldsymbol{Y}$ that maximize the likelihood. Below we focus on $P(I^i|Y^i, \boldsymbol{\Theta})$, and then define $P(Y^i|\boldsymbol{A}^i)$ and $P(\boldsymbol{A}^i)$ in section 2.4.

**Image likelihood.** The basic image units in our model are persons. Assuming independence between multiple persons in an image, the likelihood of an image can be expressed as the product over the likelihood of each person:

$$P(I^i|Y^i, \boldsymbol{\Theta}) = \prod_{I^{i,p} \in I^i} P(I^{i,p}|y^{i,p}, \boldsymbol{\Theta}) \quad (2)$$

where $y^{i,p}$ define the name-verb indices of the $p$th person in the image. A person $I^{i,p} = (I_{\text{face}}^{i,p}, I_{\text{pose}}^{i,p})$ is represented by the appearance of her face $I_{\text{face}}^{i,p}$ and pose $I_{\text{pose}}^{i,p}$. Assuming independence between the face and pose appearance of a person, the conditional probability for the appearance of the $p$th person in image $I^i$ given the latent variable $y^{i,p}$ is:

$$P(I^{i,p}|y^{i,p}, \boldsymbol{\Theta}) = P(I_{\text{face}}^{i,p}|y_{\text{face}}^{i,p}, \theta_{\text{name}}) P(I_{\text{pose}}^{i,p}|y_{\text{pose}}^{i,p}, \theta_{\text{verb}}) \quad (3)$$

where $\boldsymbol{\Theta} = (\theta_{\text{name}}, \theta_{\text{verb}})$ are the appearance models associated with the various names and verbs. Each $\theta_{\text{verb}}^v$ in $\theta_{\text{verb}} = (\theta_{\text{verb}}^1, \dots, \theta_{\text{verb}}^V, \beta_{\text{verb}})$ is a set of representative vectors modeling the variability within the pose class corresponding to a verb $v$. For example, the verb "serve" in tennis could correspond to different poses such as holding the ball on the racket, tossing the ball and hitting it. Analogously, $\theta_{\text{name}}^u$ models the variability within the face class corresponding to a name $u$.

## 2.2 Face and pose descriptors and similarity measures

After detecting faces from the images with the multi-view algorithm [23], we use [12] to detect nine distinctive feature points within the face bounding box (figure 3(left)). Each feature is represented by SIFT descriptors [18], and their concatenation gives the overall descriptor vector for the face. We use the cosine as a naturally normalized similarity measure between two face descriptors: $\text{sim}_{\text{face}}(a, b) = \frac{a^T b}{\|a\| \|b\|}$. The distance between two faces is $\text{dist}_{\text{face}}(a, b) = 1 - \text{sim}_{\cos}(a, b)$.

We use [14] to detect upper-bodies and [11] to estimate their pose. A pose $E$ consists of a distribution over the position ($x, y$ and orientation) for each of 6 body parts (head, torso, upper/lower

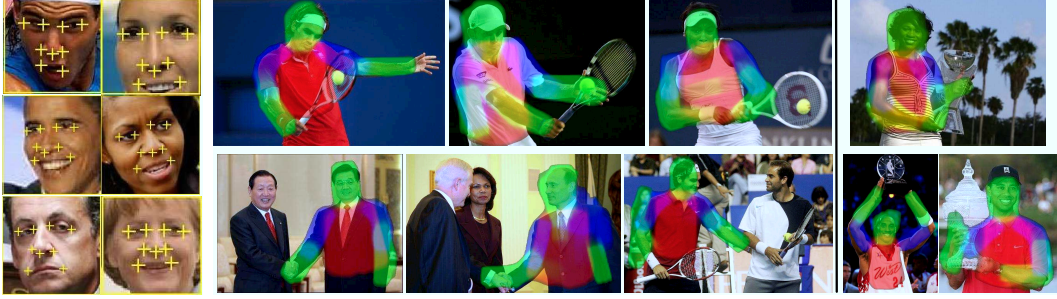

Figure 3: Example images with facial features and pose estimates superimposed. **Left** Facial features (left and right corners of each eye, two nostrils, tip of the nose, and the left and right corners of the mouth) located using [12] in the detected face bounding-box. **Right** Example estimated poses corresponding to verbs: "hit backhand", "shake hands" and "hold". Red indicates torso, blue upper arms, green lower arms and head. Brighter pixels are more likely to belong to a part. Color planes are added up, so that yellow indicates overlap between lower-arm and torso, purple between upper-arm and torso, and so on (best viewed in color).

left/right arms). The pose estimator factors out variations due to clothing and background, so $E$ conveys purely spatial arrangements of body parts. We derive three relatively low-dimensional pose descriptors from $E$, as proposed in [13]. These descriptors represent pose in different ways, such as the relative position between pairs of body parts, and part-specific soft-segmentations of the image (i.e. the probability of pixels as belonging to a part). We refer to [13, 11] for more details and the similarity measure associated with each descriptor. We normalize the range of each similarity to $[0, 1]$, and denote their average as $\mathrm{sim}_{\mathrm{pose}}(a, b)$. The final distance between two poses $a, b$ used in the rest of this paper is $\mathrm{dist}_{\mathrm{pose}}(a, b) = 1 - \mathrm{sim}_{\mathrm{pose}}(a, b)$.

## 2.3 Appearance model

The appearance model for a pose class (corresponding to a verb) is defined as:

$$P(I_{\mathrm{pose}}^{i,p}|y_{\mathrm{pose}}^{i,p}, \theta_{\mathrm{verb}}) = \sum_{k \in \{1, \ldots, V, null\}} \delta(y_{\mathrm{pose}}^{i,p}, k) \cdot P(I_{\mathrm{pose}}^{i,p}|\theta_{\mathrm{verb}}^k) \tag{4}$$

where $\theta_{\mathrm{verb}}^k$ are the parameters of the $k$th pose class (or $\beta_{\mathrm{verb}}$ if $k = null$). The indicator function $\delta(y_{\mathrm{pose}}^{i,p}, k) = 1$ if $y_{\mathrm{pose}}^{i,p} = k$ and $\delta(y_{\mathrm{pose}}^{i,p}, k) = 0$ otherwise. We only explain here the model for a pose class, as the face model is derived analogously.

How to model the conditional probability $P(I_{\mathrm{pose}}^{i,p}|\theta_{\mathrm{verb}}^k)$ is a key ingredient for the success of our approach. Some previous works on names-faces used a Gaussian mixture model [6, 21]: each name is associated with a Gaussian density, plus an additional Gaussian to model the *null* class. Using functions of the exponential family like a Gaussian simplifies computations. However, a Gaussian may restrict the representative power of the appearance model. Problems such as face and pose recognition are particularly challenging because they involve complex non-Gaussian multimodal distributions. Figure 3(right) shows a few examples of the variance within the pose class for a verb. Moreover, we cannot easily employ existing pose similarity measures [13]. Therefore, we represent the conditional probability using a exemplar-based likelihood function:

$$P(I_{\mathrm{pose}}^{i,p}|\theta_{\mathrm{verb}}^k) = \begin{cases} \frac{1}{Z_{\theta_{\mathrm{verb}}}} e^{-d_{\mathrm{pose}}(I_{\mathrm{pose}}^{i,p}, \theta_{\mathrm{verb}}^k)} & \text{if } k \in \{\text{known verbs}\} \\ \frac{1}{Z_{\theta_{\mathrm{verb}}}} e^{-\beta_{\mathrm{verb}}} & \text{if } k = null \end{cases} \tag{5}$$

where $Z_{\theta_{\mathrm{verb}}}$ is the normalizer and $d_{\mathrm{pose}}$ is the distance between the pose descriptor $I_{\mathrm{pose}}^{i,p}$ and its closest class representative vector $\mu_r^k \in \theta_{\mathrm{verb}}^k = \{\mu_{\mathrm{pose}}^{k,1}, \ldots, \mu_{\mathrm{pose}}^{k,\mathrm{R}^k}\}$, where $\mathrm{R}^k$ is the number of representative poses for verb $k$. The likelihood depends on the model parameters $\theta_{\mathrm{verb}}^k$, and the distance function $d_{\mathrm{pose}}$. The scalar $\beta_{\mathrm{verb}}$ represents the *null* model, thus poses assigned to *null* have likelihood $\frac{1}{Z_{\theta_{\mathrm{verb}}}} e^{-\beta_{\mathrm{verb}}}$. It is important to have this *null* model, as some detected persons might not correspond to any verb in the caption or they might be false detections. By generalizing the similarity measure $\mathrm{sim}_{\mathrm{pose}}(a, b)$ as a kernel product $K(a, b) = \phi(a) \cdot \phi(b)$, the distance from a vector $a$ to the sample center vector $\mu_r^k$ can be written similarly as in the weighted kernel k-means method [9]:

$$\left\| \phi(a) - \frac{\Sigma_{b \in \pi_r^k} w(b)\phi(b)}{\Sigma_{b \in \pi_r^k} w(b)} \right\|^2 = K(a, a) - \frac{2\Sigma_{b \in \pi_r^k} w(b)k(a, b)}{\Sigma_{b \in \pi_r^k} w(b)} + \frac{\Sigma_{b,d \in \pi_r^k} w(b)w(d)k(b, d)}{(\Sigma_{b \in \pi_r^k} w(b))^2} \tag{6}$$

The center vector $\mu_r^k$ is defined as $\left(\Sigma_{b \in \boldsymbol{\pi}_r^k} w(b)\phi(b)\right)/\left(\Sigma_{b \in \boldsymbol{\pi}_r^k} w(b)\right)$, where $\boldsymbol{\pi}_r^k$ is the cluster of vectors assigned to $\mu_r^k$, and $w(b)$ is the weight for each point $b$, representing the likelihood that $b$ belongs to the class of $\mu_r^k$ (as in equation (11)). This formulation can be considered as a modified version of the k-means [19] clustering algorithm. The number of centers $R^k$ can vary for different verbs, depending on the distribution of the data and the number of samples. As we are interested only in computing the distance between $\mu_r^k$ and each data point, and not in the explicit value of $\mu_r^k$, the only term that needs to be computed in equation (6) is the second (the third term is constant for each assigned $\mu_r^k$).

## 2.4 Name-verb assignments

The name-verb pairs $\boldsymbol{n}^i$ for a document are observed in its caption $C^i$. We derive from them the set of all possible assignments $\boldsymbol{A}^i = \{a_1^i, \ldots, a_{L_i}^i\}$ of name-verb pairs to persons in the image. The number of possible assignments $L_i$ depends both on the number of persons and of name-verb pairs. As opposed to the standard matching problem, here the assignments have to take into account *null*. Moreover, we have the same constraints as in the name-face problem [6]: a person can be assigned to at most one name-verb pair, and vice-versa. Therefore, given a document with $P^i$ persons and $W^i$ name-verb pairs, the number of possible assignments is $L^i = \sum_{j=0}^{\min(P^i, W^i)} \binom{P^i}{j} \cdot \binom{W^i}{j}$, where $j$ is the number of persons assigned to a name-verb pair instead of *null*. Even by imposing the above constraints, this number grows rapidly with $P^i$ and $W^i$. However, since different assignments share many common sub-assignments, the number of unique likelihood computations is much lower, namely $P^i \cdot (W^i + 1)$. Thus, we can evaluate all possible assignments for an image efficiently. Although certain assignments are unlikely to happen (e.g. all persons are assigned to *null*), here we use an uniform prior over all assignments, i.e. $P(a_l^i) = 1/L^i$. Since the true assignment $Y^i$ can only come from $\boldsymbol{A}^i$, we define the conditional probability over the latent variables $Y^i$ as:

$$P(Y^i|\boldsymbol{A}^i) = \begin{cases} 1/L^i & \text{if } Y^i \in \boldsymbol{A}^i \\ 0 & \text{otherwise} \end{cases} \tag{7}$$

The latent assignment $Y^i$ play the role of the annotations necessary for learning appearance models.

## 3 Learning the model

The task of learning is to find the model parameters $\boldsymbol{\Theta}$ and the assignments $\boldsymbol{Y}$ which maximize the likelihood of the complete dataset $\{\boldsymbol{I}, \boldsymbol{Y}, \mathcal{A}\}$. The joint probability of $\{\boldsymbol{I}, \boldsymbol{Y}, \mathcal{A}\}$ given $\boldsymbol{\Theta}$ from equation (1) can be written as

$$P(\boldsymbol{I}, \boldsymbol{Y}, \mathcal{A}|\boldsymbol{\Theta}) = \prod_{i=1}^{M} \left( P(Y^i|\boldsymbol{A}^i)P(\boldsymbol{A}^i) \prod_{p=1}^{P^i} P(I_{\text{face}}^{i,p}|y_{\text{face}}^{i,p}, \theta_{\text{name}})P(I_{\text{pose}}^{i,p}|y_{\text{pose}}^{i,p}, \theta_{\text{verb}}) \right) \tag{8}$$

Maximizing the log of this joint likelihood is equivalent to minimizing the following clustering objective function over the latent variables $\boldsymbol{Y}$ and parameters $\boldsymbol{\Theta}$:

$$\begin{aligned} \mathcal{J} = &\sum_{i,p,y_{\text{face}}^{i,p} \neq \text{null}} d_{\text{face}}(I_{\text{face}}^{i,p}, \theta_{\text{name}}^{y_{\text{face}}^{i,p}}) + \sum_{i,p,y_{\text{face}}^{i,p} = \text{null}} \beta_{\text{name}} + \sum_{i,p,y_{\text{pose}}^{i,p} \neq \text{null}} d_{\text{pose}}(I_{\text{pose}}^{i,p}, \theta_{\text{verb}}^{y_{\text{pose}}^{i,p}}) \\ &+ \sum_{i,p,y_{\text{pose}}^{i,p} = \text{null}} \beta_{\text{verb}} - \sum_i (\log P(Y^i|\boldsymbol{A}^i) + \log P(\boldsymbol{A}^i)) + \sum_{i,p} (\log Z_{\theta_{\text{name}}} + \log Z_{\theta_{\text{verb}}}) \end{aligned} \tag{9}$$

Thus, to minimize $\mathcal{J}$, each latent variable $Y^i$ must belong to the set of possible assignments $\boldsymbol{A}^i$. If $\boldsymbol{Y}$ would be known, the cluster centers $\mu \in \theta_{\text{name}}, \mu \in \theta_{\text{verb}}$ which minimize $\mathcal{J}$ could be determined uniquely (given also the number of class centers $R$). However, it is difficult to set $R$ before seeing the data. In our implementation, we determine the centers approximately using the data points and their $K$ nearest neighbors. Since estimating the normalization constants $Z_{\theta_{\text{name}}}$ and $Z_{\theta_{\text{verb}}}$ is computationally expensive, we make an approximation by considering them as constant in the clustering process (i.e. drop their terms from $\mathcal{J}$). In our experiments, this did not significantly affect the results, as also noted in several other works (e.g. [4]).

Since the assignments $\boldsymbol{Y}$ are unknown, we use a generalized EM procedure [7, 22] for simultaneously learning the parameters $\boldsymbol{\Theta}$ and solving the correspondence problem (i.e. find $\boldsymbol{Y}$):

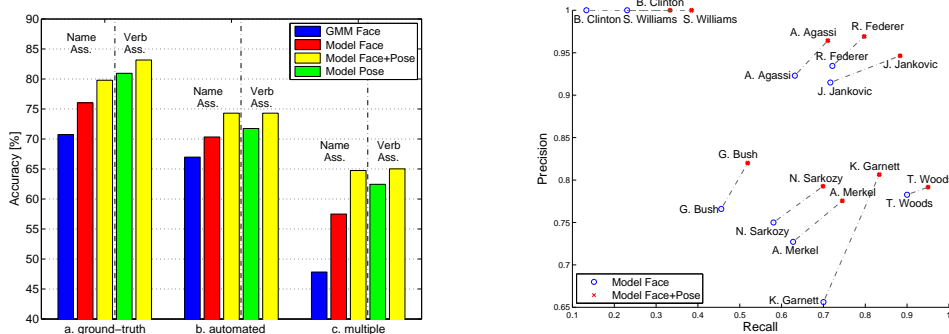

Figure 4: **Left**. Comparison of different models under different setups: using the manually annotated name-verb pairs (*ground-truth*); using the Named Entity detector and language parser (*automated*); and using the more difficult subset (*multiple*). The accuracy for name (Name Ass.) and verb (Verb Ass.) assignments are reported separately. *GMM Face* refers to the face-only model using GMM appearance models, as in [6]. **Right**. Comparison of precision and recall for 10 individuals using the stripped-down face only model, and our face+pose model. The reported results are based on automatically parsed captions for learning.

**Input.**    Data $D$; hyper-parameters $\beta_{\text{name}}$, $\beta_{\text{verb}}$, $K$

**1. Initialization.**    We start by computing the distance matrix between faces/poses from images sharing some name/verb in the caption. Next we initialize $\Theta$ using all documents in $D$. For each different name/verb, we select all captions containing only this name/verb. If the corresponding images contain only one person, their faces/poses are used to initialize the center vectors $\theta^k_{\text{name}}/\theta^k_{\text{verb}}$. The center vectors are found approximately using each data point and their $K$ nearest neighbors of the same name/verb class. If a name/verb only appears in captions with multiple names/verbs or if the corresponding images always contain multiple persons (e.g. verbs like "shake hand"), we randomly assign the name/verb to any face/pose in each image. The center vectors are then initialized using these data points. The initial weights $w$ for all data points are set to one (equation 6).

This step yields an initial estimate of the model parameters $\Theta$. We refine the parameters and assignments by repeating the following EM-steps until convergence.

**2. E-step.**    Compute the labels $Y$ using the parameters $\Theta^{old}$ from the previous iteration

$$\arg\max_Y P(Y|I, \mathcal{A}, \Theta^{old}) \propto \arg\max_Y P(I|Y, \Theta^{old})P(Y|\mathcal{A}) \qquad (10)$$

**3. M-step.**    Given the labels $Y$, update $\Theta$ so as to minimize $\mathcal{J}$ (i.e. update the cluster centers $\mu$). Our algorithm assigns each point to exactly one cluster. Each point $I^{i,p}$ in a cluster is given a weight

$$w^{i,p}_{Y^i} = \frac{P(Y^i|I^{i,p}, A^i, \Theta)}{\sum_{Y^j \in A^i} P(Y^j|I^{i,p}, A^i, \Theta)} \qquad (11)$$

which represents the likelihood that $I^{i,p}_{\text{face}}$ and $I^{i,p}_{\text{pose}}$ belong to the name and verb defined by $Y^i$. Therefore, faces and poses from images with many detections have a lower weights and contribute less to the cluster centers, reflecting the larger uncertainty in their assignments.

## 4   Experiments and conclusions

**Datasets**    There are datasets of news image-caption pairs such as those in [6, 16]. Unfortunately, these datasets are not suitable in our scenario for two reasons. Faces often occupy most of the image so the body pose is not visible. Second, the captions frequently describe the event at an abstract level, rather than using a verb to describe the actions of the persons in the image (compare figure 1 to the figures in [6, 16]). Therefore, we collected a new dataset [2] by querying Google-images using a combination of names and verbs (from sports and social interactions), corresponding to distinct upper body poses. An example query is "Barack Obama" + "shake hands". Our dataset contains 1610 images, each with at least one person whose face occupies less than 5% of the image, and with the accompanying snippet of text returned by Google-images. External annotators were asked to

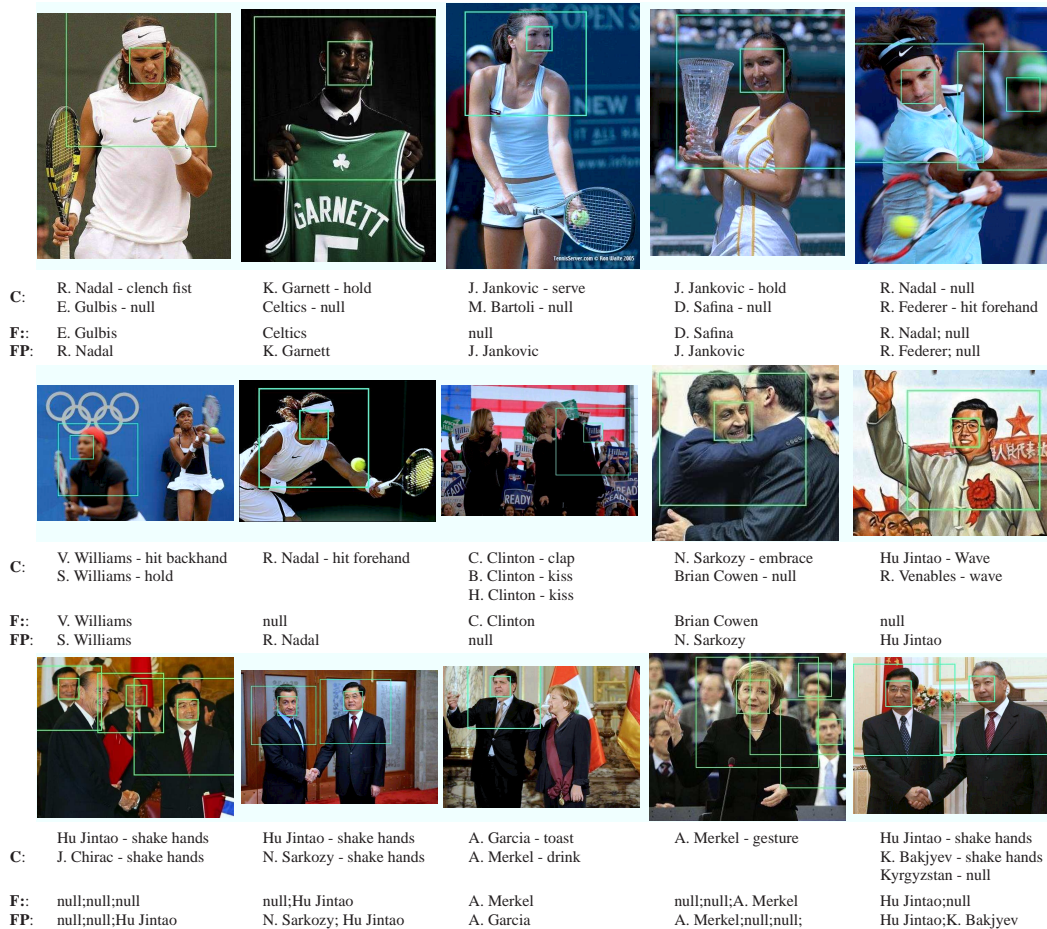

|  | | | | |
|---|---|---|---|---|
| **C**: | R. Nadal - clench fist<br>E. Gulbis - null | K. Garnett - hold<br>Celtics - null | J. Jankovic - serve<br>M. Bartoli - null | J. Jankovic - hold<br>D. Safina - null | R. Nadal - null<br>R. Federer - hit forehand |
| **F:**:<br>**FP**: | E. Gulbis<br>R. Nadal | Celtics<br>K. Garnett | null<br>J. Jankovic | D. Safina<br>J. Jankovic | R. Nadal; null<br>R. Federer; null |

|  | | | | |
|---|---|---|---|---|
| **C**: | V. Williams - hit backhand<br>S. Williams - hold | R. Nadal - hit forehand | C. Clinton - clap<br>B. Clinton - kiss<br>H. Clinton - kiss | N. Sarkozy - embrace<br>Brian Cowen - null | Hu Jintao - Wave<br>R. Venables - wave |
| **F:**:<br>**FP**: | V. Williams<br>S. Williams | null<br>R. Nadal | C. Clinton<br>null | Brian Cowen<br>N. Sarkozy | null<br>Hu Jintao |

|  | | | | |
|---|---|---|---|---|
| **C**: | Hu Jintao - shake hands<br>J. Chirac - shake hands | Hu Jintao - shake hands<br>N. Sarkozy - shake hands | A. Garcia - toast<br>A. Merkel - drink | A. Merkel - gesture | Hu Jintao - shake hands<br>K. Bakjyev - shake hands<br>Kyrgyzstan - null |
| **F:**:<br>**FP**: | null;null;null<br>null;null;Hu Jintao | null;Hu Jintao<br>N. Sarkozy; Hu Jintao | A. Merkel<br>A. Garcia | null;null;A. Merkel<br>A. Merkel;null;null; | Hu Jintao;null<br>Hu Jintao;K. Bakjyev |

Figure 5: Examples of when modeling pose improves the results at learning time. Below the images we report the name-verb pairs (**C**) from the caption as returned by the automatic parser and compare the association recovered by a model using only faces (**F**) and using both faces and poses (**FP**). The assigned names (left to right) correspond to the detected face bounding-boxes (left to right).

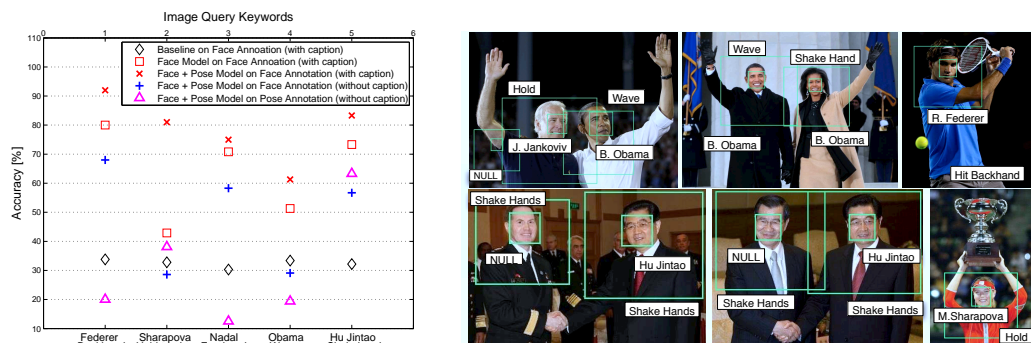

Figure 6: Recognition results on images without text captions (using models learned from automatically parsed captions). **Left** compares face annotation using different models and scenarios (see main text); **Right** shows a few examples of the labels predicted by the joint face and pose model (without using captions).

extend these snippets into realistic captions when necessary, with varied long sentences, mentioning the action of the persons in the image as well as names/verbs not appearing in the image (as 'noise', figure 1). Moreover, they also annotated the ground-truth name-verb pairs mentioned in the captions as well as the location of the target persons in the images, enabling to evaluate results quantitatively. In total the ground-truth consists of 2627 name-verb pairs. In our experiments we only consider

names and verbs occurring in at least 3 captions for a name, and 20 captions for a verb. This leaves 69 names corresponding to 69 face classes and 20 verbs corresponding to 20 pose classes.

We used an open source Named Entity recognizer [1] to detect names in the captions and a language parser [8] to find name-verbs pairs (or name-*null* if the language parser could not find a verb associated with a name). By using simple stemming rules, the same verb under different tenses and possessive adjectives was merged together. For instance "shake their hands", "is shaking hands" and "shakes hands" all correspond to the action verb "shake hands". In total, the algorithms achieves precision $85.5\%$ and recall $68.8\%$ on our dataset over the ground-truth name-verb pair. By discarding infrequent names and verbs as explained above, we retain 85 names and 20 verbs to be learned by our model (recall that some of these are false positives rather than actual person names and verbs).

**Results for learning** The learning algorithm takes about five iterations to converge. We compare experimentally our face and pose model to stripped-down versions using only face or pose information. For comparison, we also implement the constrained mixture model [6] described in section 2.3. Although [6] also originally incorporates also a language model of the caption, we discard it here so that both methods use the same amount of information. We run the experiments in three setups: (a) using the ground-truth name-verb annotations from the captions; (b) using the name-verb pairs automatically extracted by the language parser; (c) similar as (b) but only on documents with multiple persons in the image or multiple name-verb pairs in the caption. These setups are progressively more difficult, as (b) has more noisy name-verb pairs, and (c) has no documents with a single name and person, where our initialization is very reliable.

Figure 4(left) compares the accuracy achieved by different models on these setups. The accuracy is defined as the percentage of correct assignments over all detected persons, including assignments to *null*, as in [5, 16]. As the figure shows, our joint 'face and pose' model outperforms both models using face or pose alone in all setups. Both the annotation of faces *and* poses improve, demonstrating they *help each other* when successfully integrated by our model. This is the main point of the paper. Figure 4(right) shows improvements on precision and recall over models using faces or poses alone. As a second point, our model with face alone also outperforms the baseline approach using Gaussian mixture appearance models (e.g. used in [6]). Figure 5 shows a few examples of how including pose improves the learning results and solve some of the correspondence ambiguities. Improvements happen mainly in three situations: (a) when there are multiple names in a caption, as not all names in the captions are associated to action verbs (figure 1(a) and figure 5(top)); (b) when there are multiple persons in an image, because the pose disambiguates the assignment (figure 1(b) and figure 5(bottom)) and (c) when there are false detections, rare faces or faces at viewpoints different than frontal (i.e. where face recognition works less well, e.g. figure 5(middle)).

**Results for recognition** Once the model is learned, we can use it to recognize "who's doing what" in novel images with or without captions. We collected a new set of 100 images and captions from Google-images using five keywords based on names and verbs from the training dataset. We evaluate the learned model in two scenarios: (a) the test data consists of images and captions. Here we run inference on the model, recovering the best assignment $\mathbf{Y}$ from the set of possible assignments generated from the captions; (b) the same test images are used but the captions are not given, so the problem degenerates to a standard face and pose recognition task. Figure 6(left) reports face annotation accuracy for three methods using captions (scenario (a)): ($\diamond$) a baseline which randomly assigns a name (or *null*) from the caption to each face in the image; (x) our face and pose model; ($\square$) our model using only faces. The figure also shows results for scenario (b), where our full model tries to recognize faces (+) and poses ($\triangle$) in the test images without captions. On scenario (a) all models outperform the baseline, and our joint face and pose model improves significantly on the face-only model for all keywords, especially when there are multiple persons in the image.

**Conclusions.** We present an approach for the joint modeling of faces and poses in images and their association to names and action verbs in accompanying text captions. Experimental results show that our joint model performs better than face-only models both in solving the image-caption correspondence problem on the training data, and in annotating new images. Future work aims at incorporating an effective web crawler and html/language parsing tools to harvest image-caption pairs from the internet fully automatically. Other techniques such as learning distance functions [4, 15, 20] may also be incorporated during learning to improve recognition results.

**Acknowledgments** We thank K. Deschacht and M.F. Moens for providing the language parser. L. J. and B. Caputo were supported by EU project DIRAC IST-027787 and V. Ferrari by the Swiss National Science Found.

## Footnotes

[1] www.daylife.com, news.yahoo.com, news.google.com

[2] We released this dataset online at `http://www.vision.ee.ethz.ch/~ferrari`

# References

[1] http://opennlp.sourceforge.net/.

[2] K. Barnard, P. Duygulu, D. Forsyth, N. de Freitas, D. Blei, and M. Jordan. Matching words and pictures. *JMLR*, 3:1107–1135, 2003.

[3] K. Barnard and Q. Fan. Reducing correspondence ambiguity in loosely labeled training data. In *Proc. CVPR'07*.

[4] S. Basu, M. Bilenko, A. Banerjee, and R. J. Mooney. Probabilistic semi-supervised clustering with constraints. In O. Chapelle, B. Schölkopf, and A. Zien, editors, *Semi-Supervised Learning*, pages 71–98. MIT Press, 2006.

[5] T. Berg, A. Berg, J. Edwards, and D. Forsyth. Names and faces in the news. In *Proc. CVPR'04*.

[6] T. Berg, A. Berg, J. Edwards, and D. Forsyth. Who's in the picture? In *Proc. NIPS'04*.

[7] A. P. Dempster, N. Laird, and D. Rubin. Maximum likelihood from incomplete data via the em algorithm. *Journal Royal Statistical Society*, 39:1–38, 1977.

[8] K. Deschacht and M.-F. Moens. Semi-supervised semantic role labeling using the latent words language model. In *Proc. EMNLP'09*.

[9] I. Dhillon, Y. Guan, and B. Kulis. Kernel k-means: spectral clustering and normalized cuts. In *Proc. KDD'04*.

[10] P. Duygulu, K. Barnard, N. de Freitas, and D. Forsyth. Object recognition as machine translation: Learning a lexicon for a fixed image vocabulary. In *Proc. ECCV'02*.

[11] M. Eichner and V. Ferrari. Better appearance models for pictorial structures. In *Proc. BMVC'09*.

[12] M. Everingham, J. Sivic, and A. Zisserman. Hello! my name is... buffy - automatic naming of characters in tv video. In *Proc. BMVC'06*.

[13] V. Ferrari, M. Marin, and A. Zisserman. Pose search: retrieving people using their pose. In *Proc. CVPR'09*.

[14] V. Ferrari, M. Marin, and A. Zisserman. Progressive search space reduction for human pose estimation. In *Proc. CVPR'08*.

[15] A. Frome, Y. Singer, and J. Malik. Image retrieval and classification using local distance functions. In *Proc. NIPS'06*.

[16] M. Guillaumin, T. Mensink, J. Verbeek, and C. Schmid. Automatic face naming with caption-based supervision. In *Proc. CVPR'08*.

[17] A. Gupta and L. Davis. Beyond nouns: Exploiting prepositions and comparative adjectives for learning visual classifiers. In *Proc. ECCV'08*.

[18] D. Lowe. Distinctive image features from scale-invariant keypoints. *IJCV*, 60(2):91–110, 2004.

[19] J. B. MacQueen. Some methods for classification and analysis of multivariate observations. In *Proc. of 5th Berkeley Symposium on Mathematical Statistics and Probability*, 1967.

[20] T. Malisiewicz and A. Efros. Recognition by association via learning per-exemplar distances. In *Proc. CVPR'08*.

[21] T. Mensink and J. Verbeek. Improving people search using query expansions: How friends help to find people. In *Proc. ECCV'08*.

[22] R. Neal and G. E. Hinton. A view of the em algorithm that justifies incremental, sparse, and other variants. In M. I. Jordan, editor, *Learning in Graphical Models*, pages 355–368. Kluwer Academic Publishers, 1998.

[23] Y. Rodriguez. *Face Detection and Verification using Local Binary Patterns*. PhD thesis, École Polytechnique Fédérale de Lausanne, 2006.

[24] N. Shental, A. Bar-Hillel, T. Hertz, and D. Weinshall. Computing gaussian mixture models with em using equivalence constraints. In *Proc. NIPS'03*.

[25] K. Wagstaff, C. Cardie, S. Rogers, and S. Schroedl. Constrained k-means clustering with background knowledge. In *Proc. ICML'01*.

